# Fast Discriminative Visual Codebooks using Randomized Clustering Forests

**Frank Moosmann,**[*] **Bill Triggs and Frederic Jurie**
GRAVIR-CNRS-INRIA, 655 avenue de l'Europe, Montbonnot 38330, France
`firstname.lastname@inrialpes.fr`

## Abstract

Some of the most effective recent methods for content-based image classification work by extracting dense or sparse local image descriptors, quantizing them according to a coding rule such as k-means vector quantization, accumulating histograms of the resulting "visual word" codes over the image, and classifying these with a conventional classifier such as an SVM. Large numbers of descriptors and large codebooks are needed for good results and this becomes slow using k-means. We introduce *Extremely Randomized Clustering Forests* – ensembles of randomly created clustering trees – and show that these provide more accurate results, much faster training and testing and good resistance to background clutter in several state-of-the-art image classification tasks.

## 1 Introduction

Many of the most popular current methods for image classification represent images as collections of independent patches characterized by local visual descriptors. Patches can be sampled densely [18, 24], randomly [15], or at selected salient points [14]. Various local descriptors exist with different degrees of geometric and photometric invariance, but all encode the local patch appearance as a numerical vector and the more discriminant ones tend to be high-dimensional. The usual way to handle the resulting set of descriptor vectors is to vector quantize them to produce so-called *textons* [12] or *visual words* [5, 22]. The introduction of such *visual codebooks* has allowed significant advances in image classification, especially when combined with *bag-of-words* models inspired by text analysis [5, 7, 22, 24, 25].

There are various methods for creating visual codebooks. K-means clustering is currently the most common [5, 22] but mean-shift [9] and hierarchical k-means [17] clusterers have some advantages. These methods are generative but some recent approaches focus on building more discriminative codebooks [20, 24].

The above methods give impressive results but they are computationally expensive owing to the cost of assigning visual descriptors to visual words during training and use. Tree based coders [11, 17, 23] are quicker but (so far) somewhat less discriminative. It seems to be difficult to achieve both speed and good discrimination.

This paper contributes two main ideas. One is that (small) ensembles of trees eliminate many of the disadvantages of single tree based coders without losing the speed advantages of trees. The second is that classification trees contain a lot of valuable information about locality in descriptor space that is not apparent in the final class labels. One can exploit this by training them for classification then ignoring the class labels and using them as "*clustering trees*" – simple spatial partitioners that assign a distinct region label (visual word) to each leaf. Combining these ideas, we introduce *Extremely*

---

[*] Current address: Institute of Measurement and Control, University of Karlsruhe, Germany. Contact: moosmann@mrt.uka.de

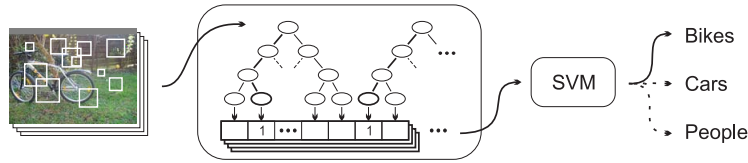

Figure 1: Using ERC-Forests as visual codebooks in bag-of-feature image classification.

*Randomized Clustering Forests* (ERC-Forests) – ensembles of randomly created clustering trees. We show that these have good resistance to background clutter and that they provide much faster training and testing and more accurate results than conventional k-means in several state-of-the-art image classification tasks.

In the rest of the paper, we first explain how decision trees can provide good visual vocabularies, then we describe our approach and present experimental results and conclusions.

## 2   Tree Structured Visual Dictionaries

Our overall goal is to classify images according to the object classes that they contain (see figure 1). We will do this by selecting or sampling patches from the image, characterizing them by vectors of local visual descriptors and coding (quantizing) the vectors using a learned *visual dictionary*, i.e. a process that assigns discrete labels to descriptors, with similar descriptors having a high probability of being assigned the same label. As in text categorization, the occurrences of each label ("visual word") are then counted to build a global histogram ("bag of words") summarizing the image ("document") contents. The histogram is fed to a classifier to estimate the image's category label. Unlike text, visual 'words' are not intrinsic entities and different quantization methods can lead to very different performances.

Computational efficiency is important because a typical image yields $10^3$–$10^4$ local descriptors and data sets often contain thousands of images. Also, many of the descriptors generally lie on the background not the object being classified, so the coding method needs to be able to learn a discriminative labelling despite considerable background 'noise'.

**K-means and tree structured codes.** Visual coding based on K-means vector quantization is effective but slow because it relies on nearest neighbor search, which remains hard to accelerate in high dimensional descriptor spaces despite years of research on spatial data structures (e.g. [21]). Nearest neighbour assignments can also be somewhat unstable: in high dimensions, concentration of measure [2] tends to ensure that there are many centres with similar distances to any given point. Component-wise decision trees offer logarithmic-time coding, but individual trees can rarely compete with a good K-means coding: each path through the tree typically accesses only a few of the feature dimensions, so there is little scope for producing a consensus over many different dimensions. Nistér *et al.*[17] introduced a tree coding based on hierarchical K-means. This uses all components and gives a good compromise between speed and loss of accuracy.

**Random forests.** Despite their popularity, we believe that K-means codes are not the best compromise. No single data structure can capture the diversity and richness of high dimensional descriptors. To do this an ensemble approach is needed. The theoretical and practical performance of ensemble classifiers is well documented [1]. Ensembles of random trees [4] seem particularly suitable for visual dictionaries owing to their simplicity, speed and performance [11]. Sufficiently diverse trees can be constructed using randomized data splits or samples [4]. *Extremely Randomized Trees* (see below) take this further by randomizing both attribute choices and quantization thresholds, obtaining even better results [8]. Compared to standard approaches such as C4.5, ER tree construction is rapid, depends only weakly on the dimensionality and requires relatively little memory.

**Clustering forests.** Methods such as [11, 8] classify descriptors by majority voting over the tree-assigned class labels. There are typically many leaves that assign a given class label. Our method works differently after the trees are built. It uses the trees as spatial partitioning methods not classifiers, assigning each leaf of each tree a distinct region label (visual word). For the overall image classification tasks studied here, histograms of these leaf labels are then accumulated over the whole

image and a global SVM classifier is applied. Our approach is thus related to *clustering trees* – decision trees whose leaves define a spatial partitioning or grouping [3, 13]. Such trees are able to find natural clusters in high dimensional spaces. They can be built without external class labels, but if labels are available they can be used to guide the tree construction. Ensemble methods and particularly forests of extremely randomized trees again offer considerable performance advantages here. The next section shows how such *Extremely Randomized Clustering Forests* can be used to produce efficient visual vocabularies for image classification tasks.

## 3 Extremely Randomized Clustering Forests (ERC-Forests)

Our goal is to build a discriminative coding method. Our method starts by building randomized decision trees that predict class labels $y$ from visual descriptor vectors $\mathbf{d} = (f_1, \ldots, f_D)$, where $f_i, i = 1, \ldots, D$ are elementary scalar features. For notational simplicity we assume that all of the descriptors from a given image share the same label $y$. We train the trees using a labeled (for now) training set $L = \{(\mathbf{d}_n, y_n), n = 1, \ldots, N\}$. However we use the trees only for spatial coding, not classification per se. During a query, for each descriptor tested, each tree is traversed from the root down to a leaf and the returned label is the unique leaf index, not the (set of) descriptor label(s) $y$ associated with the leaf.

**ERC-Trees.** The trees are built recursively top down. At each node $t$ corresponding to descriptor space region $\mathcal{R}_t$, two children $l, r$ are created by choosing a boolean test $\mathcal{T}_t$ that divides $\mathcal{R}_t$ into two disjoint regions, $\mathcal{R}_t = \mathcal{R}_l \cup \mathcal{R}_r$ with $\mathcal{R}_l \cap \mathcal{R}_r = \phi$. Recursion continues until further subdivision is impossible: either all surviving training examples belong to the same class or all have identical values for all attributes. We use thresholds on elementary features as tests, $\mathcal{T}_t = \{f_{i(t)} \leq \theta_t\}$ for some feature index $i(t)$ and threshold $\theta_t$. The tests are selected randomly as follows. A feature index $i(t)$ is chosen randomly, a threshold $\theta_t$ is sampled randomly from a uniform distribution, and the resulting node is scored over the surviving points using Shannon entropy [8]: $Sc(C, \mathcal{T}) = \frac{2I(C, \mathcal{T})}{H_C + H_\mathcal{T}}$, where $H_C$ denotes entropy of the class label distribution, $H_\mathcal{T}$ the entropy of the partition induced by the test and $I(C, \mathcal{T})$ their mutual information. High scores indicate that the split separates the classes well. This procedure is repeated until the score is higher than a fixed threshold $S_{\min}$ or until a fixed maximum number $T_{\max}$ of trials have been made. The test $\mathcal{T}_t$ that achieved the highest score is adopted and the recursion continues.

The parameters $(S_{\min}, T_{\max})$ control the strength and randomness of the generated trees. High values (e.g. $(1, D)$ for normal ID3 decision tree learning) produce highly discriminant trees with little diversity, while $S_{\min} = 0$ or $T_{\max} = 1$ produce completely random trees.

**ERC-Forests.** Compared to standard decision tree learning, the trees built using random decisions are larger and have higher variance. Class label variance can be reduced by voting over the ensemble of trees (e.g. [15]), but here, instead of voting we treat each leaf in each tree as a separate visual word and stack the leaf indices from each tree into an extended code vector for each input descriptor, leaving the integration of votes to the final classifier. The resulting process is reminiscent of spatial search algorithms based on random line projections (e.g. [10]), with each tree being responsible for distributing the data across its own set of clusters.

Classifier forests are characterized by Breiman's bound on the asymptotic generalization error [4], $PE* \leq \rho (1 - s^2)/s^2$ where $s$ measures the strength of the individual trees and $\rho$ measures the correlation between them in terms of the raw margin. It would be interesting to optimize $S_{\min}$ and $T_{\max}$ to minimize the bound but we have not yet tried this. Experimentally, the trees appear to be rather diverse while still remaining relatively strong, which should lead to good error bounds.

**Application to visual vocabularies.** In the experiments below, local features are extracted from the training images by sampling sub-windows at random positions and scales[1] and coding them using a visual descriptor function. An ERC-Forest is then built using the given class labels. To control the codebook size, we grow the trees fully then prune them back bottom up, recursively removing the node with the lowest gain until either a specified threshold on the gain or a specified number of

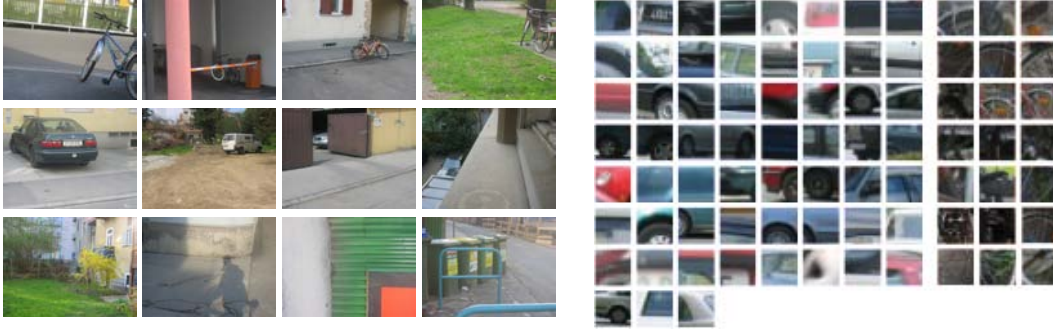

Figure 2: *Left:* example images from GRAZ-02. The rows are respectively Bikes (B), Cars (C) and background (N). *Right:* some test patches that were assigned to a particular 'car' leaf (left) and a particular 'bike' one (right)).

leaves is reached. One can also prune during construction, which is faster but does not allow the number of leaf nodes to be controlled directly.

In use, the trees transform each descriptor into a set of leaf node indices with one element from each tree. Votes for each index are accumulated into a global histogram and used for classification as in any other bag of features approach. Independently of the codebook, the denser the sampling the better the results, so typically we sample images more densely during testing than during codebook training, c.f. [18].

**Computational complexity.** The worst-case complexity for building a tree is $O(T_{\max} N k)$, where $N$ is the number of patches and $k$ is the number of clusters/leaf nodes before pruning. With adversarial data the method cannot guarantee balanced trees so it can not do better than this, but in our experiments on real data we always obtained well balanced trees at a practical complexity of around $O(T_{\max} N \log k)$. The dependence on data dimensionality $D$ is hidden in the constant $T_{\max}$, which needs to be set large enough to filter out irrelevant feature dimensions, thus providing better coding and more balanced trees. A value of $T_{\max} \sim O(\sqrt{D})$ has been suggested [8], leading to a total complexity of $O(\sqrt{D} N \log k)$. In contrast, k-means has a complexity of $O(DNk)$ which is more than $10^4$ times larger for our 768-D wavelet descriptor with $N = 20000$ image patches and $k = 5000$ clusters, not counting the number of iterations that k-means has to perform. Our method is also faster in use – a useful property given that reliable image classification requires large numbers of subwindows to be labelled [18, 24]. Labeling a descriptor with a balanced tree requires $O(\log k)$ operations whereas k-means costs $O(kD)$.

## 4   Experiments

We present detailed results on the GRAZ-02 test set, *http://www.emt.tugraz.at/˜pinz/data/*. Similar conclusions hold for two other sets that we tested, so we comment only briefly on these. GRAZ-02 (figure 2-left) contains three object categories – bicycles (B), cars (C), persons (P) – and negatives (N, meaning that none of B,C,P are present). It is challenging in the sense that the illumination is highly variable and the objects appear at a wide range of different perspectives and scales and are sometimes partially hidden. It is also neutral with respect to background, so it is not possible to detect objects reliably based on context alone.

We tested various visual descriptors. The best choice turns out to depend on the database. Our color descriptor uses raw HSL color pixels to produce a 768-D feature vector (16×16 pixels × 3 colors). Our color wavelet descriptor transforms this into another 768-D vector using a 16×16 Haar wavelet transform. Finally, we tested the popular grayscale SIFT descriptor [14], which returns 128-D vectors (4×4 histograms of 8 orientations).

We measure performance with ROC curves and classification rates at equal error rate (EER). The method is randomized so we report means and variances over 10 learning runs. We use $S_{\min} = 0.5$ but the exact value is not critical. In contrast $T_{\max}$ has a significant influence on performance so it

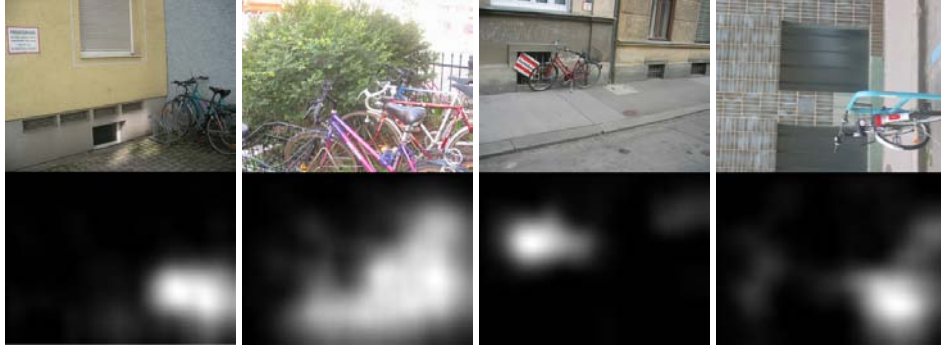

Figure 3: 'Bike' visual words for 4 different images. The brightness denotes the posterior probability for the visual word at the given image position to be labelled 'bike'.

is chosen using a validation set. For the 768-D Color Wavelet Descriptor on the GRAZ-02 dataset, $T_{max} \approx 50$.

Our algorithm's ability to produce meaningful visual words is illustrated in figure 3 (c.f. [16]). Each white dot corresponds to the center of an image sub-window that reached an unmixed leaf node for the given object category (i.e. all of the training vectors belonging to the leaf are labeled with that category). Note that even though they have been learned on entire images without object segmentation, the visual vocabulary is discriminative enough to detect local structures in the test images that correspond well with representative object fragments, as illustrated in figure 2(right).

The tests here were for individual object categories versus negatives (N). We took 300 images from each category, using images with even numbers for training and ones with odd numbers for testing. For *Setting 1* tests we trained on the whole image as in [19], while for *Setting 2* ones we used the segmentation masks provided with the images to train on the objects alone without background.

For the GRAZ-02 database the wavelet descriptors gave the best performance. We report results for these on the two hardest categories, bikes and cars. For B vs. N we achieve 84.4% average EER classification rate for setting 1 and 84.1% for setting 2, in comparison to 76.5% from Opelt *et al.*[19]. For C vs. N the respective figures are 79.9%, 79.8% and 70.7%. Remarkably, using segmentation masks during training does not improve the image classification performance. This suggests that the method is able to pick out the relevant information from a significant amount of clutter.

**Comparing ERC-Forests with k-means and kd-clustering trees.**  Unless otherwise stated, 20 000 features (67 per image) were used to learn 1000 spatial bins per tree for 5 trees, and 8000 patches were sampled per image to build the resulting 5000-D histograms. The histograms are binarized using trivial thresholding at count 1 before being fed to the global linear SVM image classifier. We also tested with histograms normalized to total sum 1, and with thresholding by maximizing the mutual information of each dimension, but neither yielded better results for ERC-Forests.

Fig. 4 gives some quantitative results on the bikes category (B vs. N). Fig. 4(a) shows the clear difference between our method and classical k-means for vocabulary construction. Note that we were not able to extend the k-means curve beyond 20 000 windows per image owing to prohibitive execution times. The figure also shows results for 'unsupervised trees' – ERC-Forests built without using the class labels during tree construction. The algorithm remains the same, but the node scoring function is defined as the ratio between the splits so as to encourage balanced trees similar to randomized KD-trees. If only a few patches are sampled this is as good as k-means and much faster. However the spatial partition is so bad that with additional test windows, the binarized histogram vectors become almost entirely filled with ones, so discrimination suffers. As the dotted line shows, using binarization thresholds that maximize the mutual information can fix this problem but the results are still far below ERC-Forests. This comparison clearly shows the advantages of using supervision during clustering.

Fig. 4(b) shows that codebooks with around 5000 entries (1000 per tree) suffice for good results. Fig. 4(c) shows that when the number of features used to build the codebooks is increased, the

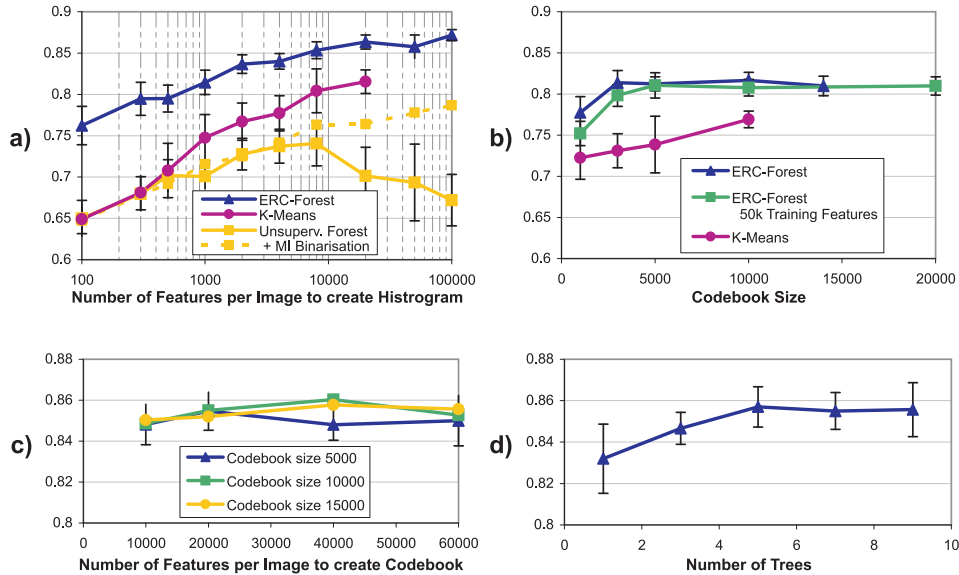

Figure 4: Evaluation of the parameters for B vs. N in setting 2: classification rate at the EER, averaged over trials. The error bars indicate standard deviations. See the text for further explanations.

optimal codebook size also increases slightly. Also, if the trees are pruned too heavily they lose discriminative power: it is better to grow them fully and do without pruning. Fig. 4(d) shows that increasing the number of trees from 1 to 5 reduces the variance and increases the accuracy, with little improvement beyond this. Here, the number of leaves per tree was kept constant at 1000, so doubling the number of trees effectively doubles the vocabulary size.

We also tested our method on the 2005 Pascal Challenge dataset, *http://www.pascal-network.org/challenges/VOC/voc2005*. This contains four categories, motorbikes, bicycles, people and cars. The goal is to distinguish each category from the others. Just 73 patches per image (50 000 in total over the 648 training images) were used to build the codebook. The maximum patch size was 30% of the image size. SIFT descriptors gave the best results for coding. The chosen forest contained four 7500-leaf trees, producing a 30 000-D histogram.

The results (M:95.8%, B:90.1%, P:94%, C:96%) were either similar to or up to 2% better than the frontrunners in the 2005 Pascal Challenge [6], but used less information and had much faster processing times. A 2.8GHz P4 took around 20 minutes to build the codebook. Building the histograms for the 684 training and 689 test images with 10 000 patches per image took only a few hours. All times include both feature extraction and coding.

We also compared our results with those of Marée *et al.* [15]. They use the same kind of tree structures to classify images directly, without introducing the vocabulary layer that we propose. Our EER error rates are consistently 5–10% better than theirs.

Finally, we tested the horse database from *http://pascal.inrialpes.fr/data/horses*. The task is difficult because the images were taken randomly from the internet and are highly variable regarding subject size, pose and visibility. Using SIFT descriptors we get an EER classification rate of 85.3%, which is significantly better than the other methods that we are aware of. 100 patches per image were used to build a codebook with 4 trees. 10 000 patches per image were used for testing.

## 5   Conclusions

Bag of local descriptor based image classifiers give state-of-the art results but require the quantization of large numbers of high-dimensional image descriptors into many label classes. *Extremely Randomized Clustering Forests* provide a rapid and highly discriminative approach to this that outperforms k-means based coding in training time and memory, testing time, and classification accuracy. The method can use unlabelled data but it benefits significantly from labels when they are

available. It is also resistant to background clutter, giving relatively clean segmentation and "pop-out" of foreground classes even when trained on images that contain significantly more background features than foreground ones. Although trained as classifiers, the trees are used as descriptor-space quantization rules with the final classification being handled by a separate SVM trained on the leaf indices. This seems to be a promising approach for visual recognition, and may be beneficial in other areas such as object detection and segmentation.

## Footnotes

[1]For image classification, dense enough random sampling eventually outperforms keypoint based sampling [18].

## References

[1] E. Bauer and R. Kohavi. An empirical comparison of voting classification algorithms: Bagging, boosting, and variants. *Machine Learning Journal*, 36(1-2):105–139, 1999.

[2] K. Beyer, J. Goldstein, R. Ramakrishnan, and U. Shaft. When is nearest neighbors meaningful? In *Int. Conf. Database Theorie*, pages 217–235, 1999.

[3] H. Blockeel, L. De Raedt, and J. Ramon. Top-down induction of clustering trees. In *ICML*, pages 55–63, 1998.

[4] L. Breiman. Random forests. *ML Journal*, 45(1):5–32, 2001.

[5] G. Csurka, C. Dance, L. Fan, J. Williamowski, and C. Bray. Visual categorization with bags of keypoints. In *ECCV'04 workshop on Statistical Learning in CV*, pages 59–74, 2004.

[6] M. Everingham et al. (33 authors). The 2005 PASCAL visual object classes challenge. In F. d'Alche Buc, I. Dagan, and J. Quinonero, editors, *Proc. 1st PASCAL Challenges Workshop*. Springer LNAI, 2006.

[7] R. Fergus, L. Fei-Fei, P. Perona, and A. Zisserman. Learning object categories from google's image search. In *ICCV*, pages II: 1816–1823, 2005.

[8] P. Geurts, D. Ernst, and L. Wehenkel. Extremely randomized trees. *Machile Learning Journal*, 63(1), 2006.

[9] F. Jurie and B. Triggs. Creating efficient codebooks for visual recognition. In *ICCV*, 2005.

[10] H. Lejsek, F.H. Ásmundsson, B. Thór-Jónsson, and L. Amsaleg. Scalability of local image descriptors: A comparative study. In *ACM Int. Conf. on Multimedia*, Santa Barbara, 2006.

[11] V. Lepetit, P. Lagger, and P. Fua. Randomized trees for real-time keypoint recognition. In *CVPR '05 Vol.2*, pages 775–781, 2005.

[12] T. Leung and J. Malik. Representing and recognizing the visual appearance of materials using three-dimensional textons. *IJCV*, 43(1):29–44, June 2001.

[13] Bing Liu, Yiyuan Xia, and Philip S. Yu. Clustering through decision tree construction. In *CIKM '00*, pages 20–29, 2000.

[14] D.G. Lowe. Distinctive image features from scale-invariant keypoints. *IJCV*, 60(2), 2004.

[15] R. Marée, P. Geurts, J. Piater, and L. Wehenkel. Random subwindows for robust image classification. In *CVPR*, volume 1, pages 34–40, 2005.

[16] F. Moosmann, D. Larlus, and F. Jurie. Learning saliency maps for object categorization. In *ECCV'06 Workshop on the Representation and Use of Prior Knowledge in Vision*, 2006.

[17] D. Nistér and H. Stewénius. Scalable recognition with a vocabulary tree. In *CVPR*, 2006.

[18] E. Nowak, F. Jurie, and B. Triggs. Sampling strategies for bag-of-features image classification. In *ECCV'06*, 2006.

[19] A. Opelt and A. Pinz. Object localization with boosting and weak supervision for generic object recognition. In *SCIA*, 2005.

[20] F. Perronnin, C. Dance, G. Csurka, and M. Bressan. Adapted vocabularies for generic visual categorization. In *ECCV*, 2006.

[21] U. Shaft, J. Goldstein, and K. Beyer. Nearest neighbor query performance for unstable distributions. Technical Report TR 1388, Dpt of Computer Science, Univ. of Wisconsin, 1998.

[22] J. Sivic and A. Zisserman. Video Google: A text retrieval approach to object matching in videos. In *ICCV*, volume 2, pages 1470–1477, October 2003.

[23] J. Winn and A. Criminisi. Object class recognition at a glance. In *CVPR'06 - video tracks*, 2006.

[24] J. Winn, A. Criminisi, and T. Minka. Object categorization by learned universal visual dictionary. In *ICCV*, pages II: 1800–1807, 2005.

[25] J. Zhang, M. Marszalek, S. Lazebnik, and C. Schmid. Local features and kernels for classification of texture and object categories: A comprehensive study. *Int. J. Computer Vision. To appear*, 2006.
